# Training fMRI Classifiers to Discriminate Cognitive States across Multiple Subjects

**Xuerui Wang, Rebecca Hutchinson, and Tom M. Mitchell**
Center for Automated Learning and Discovery
Carnegie Mellon University
5000 Forbes Avenue, Pittsburgh, PA 15213
{xuerui.wang, rebecca.hutchinson, tom.mitchell}@cs.cmu.edu

## Abstract

We consider learning to classify cognitive states of human subjects, based on their brain activity observed via functional Magnetic Resonance Imaging (fMRI). This problem is important because such classifiers constitute "virtual sensors" of hidden cognitive states, which may be useful in cognitive science research and clinical applications. In recent work, Mitchell, et al. [6,7,9] have demonstrated the feasibility of training such classifiers for individual human subjects (e.g., to distinguish whether the subject is reading an ambiguous or unambiguous sentence, or whether they are reading a noun or a verb). Here we extend that line of research, exploring how to train classifiers that can be applied across *multiple* human subjects, including subjects who were not involved in training the classifier. We describe the design of several machine learning approaches to training multiple-subject classifiers, and report experimental results demonstrating the success of these methods in learning cross-subject classifiers for two different fMRI data sets.

## 1   Introduction

The advent of functional Magnetic Resonance Imaging (fMRI) has made it possible to safely, non-invasively observe correlates of neural activity across the entire human brain at high spatial resolution. A typical fMRI session can produce a three dimensional image of brain activation once per second, with a spatial resolution of a few millimeters, yielding tens of millions of individual fMRI observations over the course of a twenty-minute session. This fMRI technology holds the potential to revolutionize studies of human cognitive processing, provided we can develop appropriate data analysis methods.

Researchers have now employed fMRI to conduct hundreds of studies that identify which regions of the brain are activated *on average* when a human performs a particular cognitive task (e.g., reading, puzzle solving). Typical research publications describe summary statistics of brain activity in various locations, calculated by averaging together fMRI observations collected over multiple time intervals during which the subject responds to repeated stimuli of a particular type.

Our interest here is in a different problem: training classifiers to automatically decode the

subject's cognitive state at a *single instant or interval in time*. If we can reliably train such classifiers, we may be able to use these as "virtual sensors" of hidden cognitive states, to observe previously hidden cognitive processes in the brain.

In recent work [6,7,9], Mitchell et al. have demonstrated the feasibility of training such classifiers. Whereas their work focussed primarily on training a different classifier for each human subject, our focus in this paper is on training a single classifier that can be used across multiple human subjects, including humans not involved in the training process. This is challenging because different brains have substantially different sizes and shapes, and because different people may generate different brain activation given the same cognitive state. Below we briefly survey related work, describe a range of machine learning approaches to this problem, and present experimental results showing statistically significant cross-subject classifier accuracies for two different fMRI studies.

## 2 Related Work

As noted above, Mitchell et al. [6,7,9] describe methods for training classifiers of cognitive states, focussing primarily on training subject-specific classifiers. More specifically, they train classifiers that distinguish among a set of predefined cognitive states, based on a single fMRI image or fixed window of fMRI images collected relative to the presentation of a particular stimulus. For example, they report on successful classifiers to distinguish whether the object presented to the subject is a sentence or a picture, whether the sentence being viewed is ambiguous or unambiguous, whether an isolated word is a noun or a verb, and whether an isolated noun is about a person, building, animal, etc. They used several different classifiers, and report that dimensionality reduction methods are essential given the high dimensional, sparse training data. They propose specific methods for dimensionality reduction that take advantage of data collected during rest periods between stimuli, and demonstrate that these outperform standard methods for feature selection such as those based on mutual information. Despite these positive results, there remain several limitations: classifiers are trained and applied over a fixed time window of data, classifiers are trained only to discriminate among predefined classes of cognitive states, and they deal only with single cognitive states rather than multiple states evolving over time.

In earlier work, Wagner et al. [11] report that they have been able to predict whether a verbal experience will be remembered later, based on the magnitude of activity within certain parts of left prefrontal and temporal cortices during that experience. Haxby et al. [2] show that different patterns of fMRI activity are generated when a subject views a photograph of a face versus a house, etc., and show that by dividing the fMRI data for each photograph category into two samples, they could automatically match the data samples related to the same category. Recent work on brain computer interfaces (see, e.g., [8]) also seeks to decode observed brain activity (often EEG or direct neural recordings, rather than fMRI) typically for the purpose of controlling external devices.

## 3 Approach

### 3.1 Learning Method

In this paper we explore the use of machine learning methods to approximate classification functions of the following form

$$f : \langle I_1, ..., I_n \rangle \rightarrow \text{CognitiveState}$$

where $\langle I_1, ..., I_n \rangle$ is a sequence of $n$ fMRI images collected during a contiguous time interval and where CognitiveState is the set of cognitive states to be discriminated. We explore a number of classifier training methods, including:

- *Gaussian Naive Bayes (GNB).* This classifier learns a class-conditional Gaussian generative model for each feature[1]. New examples are classified using Bayes rule and the assumption that features are conditionally independent given the class (see, for instance, [5]).

- *Support Vector Machine (SVM).* We employ a linear kernel Support Vector Machine (see, for instance, [1]).

- *k Nearest Neighbor(kNN).* We use $k$ Nearest Neighbor with a Euclidean distance metric, considering values of 1, 3, and 5 for $k$ (see, for instance, [5]).

Classifiers were evaluated using a "leave one subject out" cross validation procedure, in which each of the $m$ human subjects was used as a test subject while training on the remaining $m-1$ subjects, and the mean accuracy over these held out subjects was calculated.

## 3.2 Feature Extraction

In general, each input image may contain many thousands of voxels. We explored a variety of approaches to reducing the dimensionality of the input feature vector, including methods that select a subset of available features, methods that replace multiple feature values by their mean, and methods that use both of these extractions. In the latter two cases, we take means over values found within anatomically defined brain regions (e.g., dorsolateral prefrontal cortex) which are referred to as Regions of Interest, or ROIs).

We considered the following feature extraction methods:

- *Average.* For each ROI, calculate the mean activity over all voxels in the ROI. Use these ROI means as the input features.

- *ActiveAvg(n).* For each ROI, select the $n$ most active voxels[2], then calculate the mean of their values. Again, use these ROI means as the input features. Here the "most active" voxels are those whose activity while performing the task varies the most from their activity when the subject is at rest (see [7] for details).

- *Active(n).* Select the $n$ most active voxels over the entire brain. Use only these $n$ voxels as input features.

## 3.3 Registering Data from Multiple Subjects

Given the different sizes and shapes of different brains, it is not possible to directly map the voxels in one brain to those in another. We considered two different methods for producing representations of fMRI data for use across multiple subjects:

- *ROI Mapping.* Abstract the voxel data in each brain using the *Average* or *ActiveAvg(n)* feature extraction method described above. Because each brain contains the same set of anatomically defined ROIs, we can use the resulting representation of average activity per ROI as a canonical representation across subjects.

- *Talairach coordinates.* The coordinate system of each brain is transformed (geometrically morphed) into the coordinate system of a standard brain (known as the Talairach-Tournoux coordinate system [10]). After this transformation, each brain has the same shape and size, though the transformation is usually imperfect.

There are significant differences in these two approaches. First, note they differ in their spatial resolution and in the dimension of the resulting input feature vector. ROI Mapping results in just one feature per ROI (we work with at most 35 ROIs per brain) at each timepoint, whereas Talairach coordinates retain the voxel-level resolution (on the order of 15,000 voxels per brain). Second, the approaches have different noise characteristics. ROI Mapping reduces noise by averaging voxel activations, whereas the Talairach transformation effectively introduces new noise due to imperfections in the morphing transformation. Thus, the approaches have complementary advantages and disadvantages. Notice both of these transformations require background knowledge about brain anatomy in order to identify anatomical landmarks or ROIs.

## 4 Case Studies

This section describes two fMRI case studies used for training classifiers (detailed in [7]).

### 4.1 Sentence versus Picture Study

In this fMRI study [3], thirteen normal subjects performed a sequence of trials. During each trial they were first shown a sentence and a simple picture, then asked whether the sentence correctly described the picture. We used this data set to explore the feasibility of training classifiers to distinguish whether the subject is examining a sentence or a picture during a particular time interval.

In half of the trials the picture was presented first, followed by the sentence, which we will refer to as *PS data set*. In the remaining trials, the sentence was presented first, followed by the picture, which we will call *SP data set*. Pictures contained geometric arrangements of two of the following symbols: $+$, $*$, $\$$. Sentences were descriptions such as "It is true that the star is below the plus," or "It is not true that the star is above the plus."

The learning task we consider here is to train a classifier to determine, given a particular 16-image interval of fMRI data, whether the subject was viewing a sentence or a picture during this interval. In other words, we wish to learn a classifier of the form:

$$f : \langle I_1, ..., I_{16} \rangle \rightarrow \{\text{Picture, Sentence}\}$$

where $I_1$ is the image captured at the time of stimulus (picture or sentence) onset. In this case we restrict the classifier input to 7 most relevant ROIs[3] determined by a domain expert.

### 4.2 Syntactic Ambiguity Study

In this fMRI study [4], subjects were presented with ambiguous and unambiguous sentences, and were asked to respond to a yes-no question about the content of each sentence. The questions were designed to ensure that the subject was in fact processing the sentence. Five normal subjects participated in this study, which we will refer to as *SA data set*.

We are interested here in learning a classifier that takes as input an interval of fMRI activity, and determines whether the subject was currently reading an unambiguous or ambiguous sentence. An example ambiguous sentence is "The experienced soldiers warned about the dangers conducted the midnight raid." An example of an unambiguous sentence is "The experienced soldiers spoke about the dangers before the midnight raid." We train classifiers of the form

$$f : \langle I_1, ..., I_{16} \rangle \rightarrow \{\text{Ambiguous, Unambiguous}\}$$

where $I_1$ is the image captured at the time when the sentence is first presented to the subject. In this case we restrict the classifier input to 4 ROIs[4] considered to be the most relevant.

## 5 Experimental Results

The primary goal of this work is to determine whether and how it is possible to train classifiers of cognitive states across multiple human subjects. We experimented using data from the two case studies described above, measuring the accuracy of classifiers trained for single subjects, as well as those trained for multiple subjects. Note we might expect the multiple subject classification accuracies to be lower due to differences among subjects, or to be higher due to the larger number of training examples available.

In order to test the statistical significance of our results, consider the 95% confidence intervals[5] of the accuracies. Assuming that errors on test examples are i.i.d. Bernoulli($p$) distributed, the number of observed correct classifications will follow a Binomial($n, p$) distribution, where $n$ is the number of test examples. Table 1 displays the lowest accuracies that are statistically significant at the 95% confidence level, where the expected accuracy due to chance is 0.5 given the equal number of examples from both classes. We will not report confidence interval individually for each accuracy because they are very similar.

Table 1: The lowest accuracies that are significantly better than chance at the 95% level.

|  | SP | PS | SP+PS | SA |
|---|---|---|---|---|
| **# of examples** | 520 | 520 | 1040 | 100 |
| **Lowest accuracy** | 54.3% | 54.3% | 53.1% | 59.7% |

### 5.1 ROI Mapping

We first consider the ROI Mapping method for merging data from multiple subjects. Table 2 shows the classifier accuracies for the Sentence versus Picture study, when training across subjects and testing on the subject withheld from the training set. For comparison, it also shows (in parentheses) the average accuracy achieved by classifiers trained and tested on single subjects. All results are highly significant compared to the 50% accuracy expected by chance, demonstrating convincingly the feasibility of training classifiers to distinguish cognitive states in subjects beyond the training set. In fact, the accuracy achieved on the left out subject for the multiple-subject classifiers is often very close to the average accuracy of the single-subject classifiers, and in several cases it is significantly better. This surprisingly positive result indicates that the accuracy of the multiple-subject classifier, when tested on new subjects outside the training set, is comparable to the average accuracy achieved when training and testing using data from a single subject. Presumably this can be explained by the fact that it is trained using an order of magnitude more training examples, from twelve subjects rather than one. The increase in training set size apparently compensates for the variability among subjects.

A second trend apparent in Table 2 is that the accuracies in SP or PS data sets are better than the accuracies when using their union (SP+PS). Presumably this is due to the fact that the context in which the stimulus (picture or sentence) appears is more consistent when we restrict to data in which these stimuli are presented in the same sequence.

Table 2: Multiple-subject accuracies in the Sentence versus Picture study (ROI mapping). Numbers in parenthesis are the corresponding mean accuracies of single-subject classifiers.

| METHOD | CLASSIFIER | SP | PS | SP+PS |
|---|---|---|---|---|
| Average | GNB | 88.8% (90.6%) | 82.3% (79.6%) | 74.3% (66.5%) |
| Average | SVM | 86.5% (89.0%) | 77.1% (83.7%) | 75.3% (69.8%) |
| Average | 1NN | 84.8% (86.5%) | 73.8% (61.9%) | 63.7% (59.7%) |
| Average | 3NN | 86.5% (87.5%) | 75.8% (69.2%) | 67.3% (59.7%) |
| Average | 5NN | 88.7% (89.4%) | 78.7% (74.6%) | 68.3% (60.4%) |
| ActiveAvg(20) | GNB | 92.5% (95.4%) | 87.3% (88.1%) | 72.8% (75.4%) |
| ActiveAvg(20) | 1NN | 91.5% (94.4%) | 83.8% (82.5%) | 66.0% (71.2%) |
| ActiveAvg(20) | 3NN | 93.1% (95.4%) | 86.2% (83.7%) | 71.5% (73.2%) |
| ActiveAvg(20) | 5NN | 93.8% (95.0%) | 87.5% (86.2%) | 72.0% (73.2%) |

Table 3: Multiple-subject accuracies in the Syntactic Ambiguity study (ROI mapping). Numbers in parenthesis are the corresponding mean accuracies of single-subject classifiers. To choose $n$ in ActiveAvg($n$), we explored all even numbers less than 50, reporting the best.

| METHOD | CLASSIFIER | ACCURACY |
|---|---|---|
| Average | GNB | 58.0% (61.0%) |
| Average | SVM | 54.0% (63.0%) |
| Average | 1NN | 56.0% (54.0%) |
| Average | 3NN | 57.0% (64.0%) |
| Average | 5NN | 58.0% (60.0%) |
| ActiveAvg($n$) | GNB | 64.0% (68.0%) |
| ActiveAvg($n$) | SVM | 65.0% (71.0%) |
| ActiveAvg($n$) | 1NN | 64.0% (61.0%) |
| ActiveAvg($n$) | 3NN | 69.0% (60.0%) |
| ActiveAvg($n$) | 5NN | 62.0% (64.0%) |

Classifier accuracies for the Syntactic Ambiguity study are shown in Table 3. Note accuracies above 59.7% are significantly better than chance. The accuracies for both single-subject and multiple-subject classifiers are lower than in the first study, perhaps due in part to the smaller number of subjects and training examples. Although we cannot draw strong conclusions from the results of this study, it provides modest additional support for the feasibility of training multiple-subject classifiers using ROI mapping. Note that accuracies of the multiple-subject classifiers are again comparable to those of single subject classifiers.

## 5.2 Talairach Coordinates

Next we explore the Talairach coordinates method for merging data from multiple subjects. Here we consider the Syntactic Ambiguity study only[6]. Note one difficulty in utilizing the Talairach transformation here is that slightly different regions of the brain were scanned for different subjects. Figure 1 shows the portions of the brain that were scanned for two of the subjects along with the intersection of these regions from all five subjects. In combining data from multiple subjects, we used only the data in this intersection.

---

the true confidence interval of the mean accuracy, which can be shown using the Lagrangian method.

[6]We experienced technical difficulties in applying the Talairach transformation software to the Sentence versus Picture study (see [3] for details).

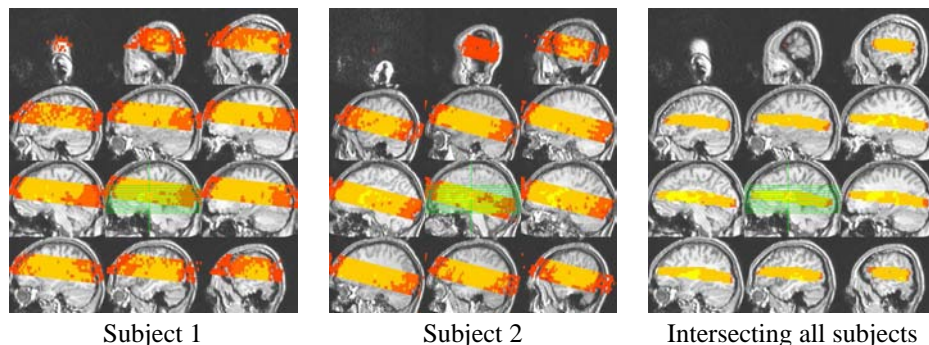

| Subject 1 | Subject 2 | Intersecting all subjects |

Figure 1: The two leftmost panels show in color the scanned portion of the brain for two subjects (Syntactic Ambiguity study) in Talairach space in sagittal view. The rightmost panel shows the intersection of these scanned bands across all five subjects.

The results of training multiple-subject classifiers based on the Talairach coordinates method are shown in Table 4. Notice the results are comparable to those achieved by the earlier ROI Mapping method in Table 3. Based on these results, we cannot state that one of these methods is significantly more accurate than the other. When using the Talairach method, we found the most effective feature extraction approach was the Active($n$) feature selection approach, which chooses the $n$ most active voxels from across the brain. Note that it is not possible to use this feature selection approach with the ROI Mapping method, because the individual voxels from different brains can only be aligned after performing the Talairach transformation.

Table 4: Multiple-subject accuracies in the Syntactic Ambiguity study (Talairach coordinates). Numbers in parenthesis are the mean accuracies of single-subject classifiers. For $n$ in Active($n$), we explored all even numbers less than 200, reporting the best.

| METHOD | CLASSIFIER | ACCURACY |
|---|---|---|
| Active($n$) | GNB | 63.0% (72.0%) |
| Active($n$) | SVM | 67.0% (71.0%) |
| Active($n$) | 1NN | 60.0% (64.0%) |
| Active($n$) | 3NN | 60.0% (69.0%) |
| Active($n$) | 5NN | 62.0% (69.0%) |

## 6   Summary and Conclusions

The primary goal of this research was to determine whether it is feasible to use machine learning methods to decode mental states across multiple human subjects. The successful results for two case studies indicate that this is indeed feasible.

Two methods were explored to train multiple-subject classifiers based on fMRI data. ROI mapping abstracts fMRI data by using the mean fMRI activity in each of several anatomically defined ROIs to map different brains in terms of ROIs. The transformation to Talairach coordinates morphs brains into a standard coordinate frame, retaining the approximate spatial resolution of the original data. Using these approaches, it was possible to train classifiers to distinguish, e.g., whether the subject was viewing a picture or a sentence describing a picture, and to apply these successfully to subjects outside the training set. In many cases, the classification accuracy for subjects outside the training set equalled or

exceeded the accuracy achieved by training on data from just the single subject. The results using the two methods showed no statistically significant difference in the Syntactic Ambiguity study.

It is important to note that while our empirical results demonstrate the ability to successfully distinguish among a predefined set of states occurring at specific times while the subject performs specific tasks, they do not yet demonstrate that trained classifiers can reliably detect cognitive states occurring at arbitrary times while the subject performs arbitrary tasks. We intend to pursue this more general goal in future work. We foresee many opportunities for future machine learning research in this area. For example, we plan to next learn models of temporal behavior, in contrast to the work reported here which considers only data at a single time interval. Machine learning methods such as Hidden Markov Models and Dynamic Bayesian Networks appear relevant. A second research direction is to develop learning methods that take advantage of data from multiple studies, in contrast to the single study efforts described here.

## Acknowledgments

We are grateful to Marcel Just for providing the fMRI data for these experiments, and for many valuable discussions and suggestions. We would like to thank Francisco Pereira and Radu S. Niculescu for providing much code to run our experiments, and Vladimir Cherkassky, Joel Welling, Erika Laing and Timothy Keller for their instruction on techniques related to Talairach transformation.

## Footnotes

[1]It is well known that the Gaussian model does not accurately fit fMRI data. Some non-Gaussian models, such as Generalized Gaussian model which makes use of the kurtosis of the data, and $t$-distribution which is more heavy-tailed, are in our future plan.

[2]The fMRI data used here are first preprocessed by FIASCO (http://www.stat.cmu.edu/~fiasco), and the active voxels are determined by $t$-test.

[3]They are pars opercularis of the inferior frontal gyrus, pars triangularis of the inferior frontal gyrus, intra-parietal sulcus, inferior temporal gyri and sulci, inferior parietal lobule, dorsolateral prefrontal cortex, and an area around the calcarine sulcus, respectively.

[4]They include pars opercularis of the inferior frontal gyrus, pars triangularis of the inferior frontal gyrus, Wernicke's area, and the superior temporal gyrus.

[5]Under cross validation, we learn $m$ classifiers, and the accuracy we reported is the mean accuracy of these classifiers. The size of the confidence interval we compute is the upper bound of the size of

## References

[1] Burges, C., A Tutorial on Support Vector Machines for Pattern Recognition, *Journal of data Mining and Knowledge Discovery*, 2(2),121-167, 1998.

[2] Haxby, J., Gobbini, M., Furey, M., Ishai, A., Schouten, J., & Pietrini, P., Distributed and Overlapping Representations of Faces and Objects in Ventral Temporal Cortex, *Science*, 293, 2425-2430, 2001.

[3] Keller, T., Just, M., & Stenger, V., Reading Span and the Time-course of Cortical Activation in Sentence-Picture Verification,*Annual Convention of the Psychonomic Society*, Orlando, FL, 2001.

[4] Mason, R., Just, M., Keller, T., & Carpenter, P., Ambiguity in the Brain: What Brain Imaging Reveals about the Processing of Syntactically Ambiguous Sentences, *Journal of Experimental Psychology: Learning, Memory, and Cognition*, in press, 2003.

[5] Mitchell, T.M., *Machine Learning*, McGraw-Hill, 1997

[6] Mitchell, T.M., Hutchinson, R., Just, M., Niculescu, R., Pereira, F., & Wang, X., Classifying Instantaneous Cognitive States from fMRI Data, *The American Medical Informatics Association 2003 Annual Symposium*, to appear, 2003

[7] Mitchell, T.M., Hutchinson, R., Niculescu, R., Pereira, F., Wang, X., Just, M., & Newman, S., Learning to Decode Cognitive States from Brain Images, *Machine Learning: Special Issue on Data Mining Lessons Learned*, accepted, 2003

[8] *NIPS 2001 Brain Computer Interface Workshop*, Whistler, BC, Canada, December 2001.

[9] Pereira, F., Just, M., & Mitchell, T.M., Distinguishing Natural Language Processes on the Basis of fMRI-measured Brain Activation, *PKDD 2001*, Freiburg, Germany, 2001.

[10] Talairach, J., & Tournoux, P., *Co-planar Stereotaxic Atlas of the Human Brain*, Thieme, New York, 1988.

[11] Wagner, A., Schacter, D., Rotte, M., Koutstaal, W., Maril, A., Dale, A., Rosen, B., & Buckner, R., Building Memories: Remembering and Forgetting of Verbal Experiences as Predicted by Brain Activity, *Science*, 281, 1188-1191, 1998.
